# Some Approximation Properties of Projection Pursuit Learning Networks

**Ying Zhao     Christopher G. Atkeson**
The Artificial Intelligence Laboratory
Massachusetts Institute of Technology
Cambridge, MA 02139

## Abstract

This paper will address an important question in machine learning: What kind of network architectures work better on what kind of problems? A projection pursuit learning network has a very similar structure to a one hidden layer sigmoidal neural network. A general method based on a continuous version of projection pursuit regression is developed to show that projection pursuit regression works better on angular smooth functions than on Laplacian smooth functions. There exists a ridge function approximation scheme to avoid the curse of dimensionality for approximating functions in $L^2(\phi_d)$.

## 1   INTRODUCTION

Projection pursuit is a nonparametric statistical technique to find "interesting" low dimensional projections of high dimensional data sets. It has been used for nonparametric fitting and other data-analytic purposes (Friedman and Stuetzle, 1981, Huber, 1985). Approximation properties have been studied by Diaconis & Shahshahani (1984) and Donoho & Johnstone (1989). It was first introduced into the context of learning networks by Barron & Barron (1988). A one hidden layer sigmoidal feedforward neural network approximates $f(\mathbf{x})$ using the structure (Figure 1(a)):

$$g(\mathbf{x}) = \sum_{j=1}^{n} \alpha_j \sigma(p_j \theta_j^T \mathbf{x} + \delta_j) \qquad (1)$$

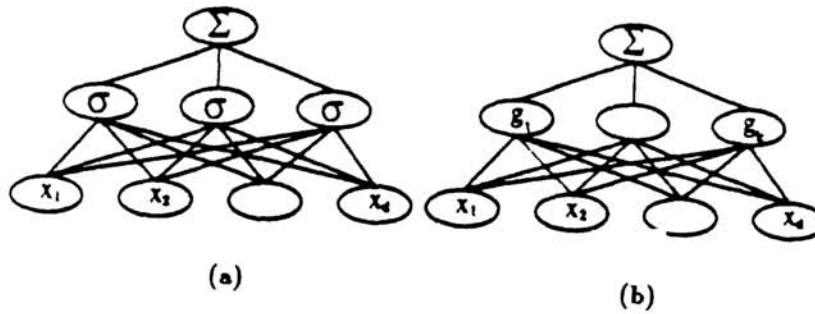

Figure 1: (a) A One Hidden Layer Feedforward Neural Network. (b) A Projection Pursuit Learning Network.

where $\sigma$ is a sigmoidal function. $\theta_j$ are direction parameters with $\|\theta_j\| = 1$, and $\alpha_j, p_j, \delta_j$ are function parameters. A projection pursuit learning network based on projection pursuit regression (PPR) (Friedman and Stuetzle, 1981) or a ridge function approximation scheme (RA) has a very similar structure (Figure 1(b)).

$$g(\mathbf{x}) = \sum_{j=1}^{n} g_j(\theta_j^T \mathbf{x}) \qquad (2)$$

where $\theta_j$ are also direction parameters with $\|\theta_j\| = 1$. The corresponding function parameters are ridge functions $g_j$ which are any smooth function to be learned from the data. Since $\sigma$ is replaced by a more general smooth function $g_j$, projection pursuit learning networks can be viewed as a generalization of one hidden layer sigmoidal feedforward neural networks. This paper will discuss some approximation properties of PPR:

1. Projection pursuit learning networks work better on angular smooth functions than on Laplacian smooth functions. Here "work better" means that for fixed complexities of hidden unit functions and a certain accuracy, fewer hidden units are required. For the two dimensional case ($d = 2$), Donoho and Johnstone (1989) show this result using equispaced directions. The equispaced directions may not be available when $d > 2$. We use a set of directions generated from zeros of orthogonal polynomials and uniformly distributed directions on an unit sphere instead. The analysis method in D & J's paper is limited to two dimensions. We apply the theory of spherical harmonics (Muller, 1966) to develop a continuous ridge function representation of any arbitray smooth functions and then employ different numerical integration schemes to discretize it for cases when $d > 2$.

2. The curse of dimensionality can be avoided when a proper ridge function approximation is applied. Once a continuous ridge function representation is established for any function in $L^2(\phi_d)$, a Monte Carlo type of integration scheme can be applied which has a RMS error convergence rate $O(N^{-\frac{1}{2}})$ where $N$ is the number of ridge functions in the linear combinations. This is a similar result to Barron's result (Barron, 1991) except that we have less restrictions on the underlying function class.

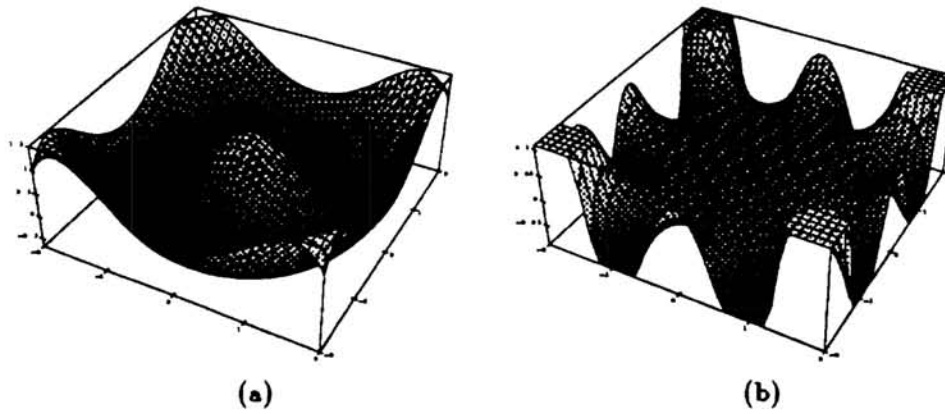

(a)                                              (b)

Figure 2: (a) A radial basis element $J_{014}$. (b) a harmonic basis element $J_{810}$.

## 2   SMOOTHNESS CLASSES AND A $L^2(\phi_d)$ BASIS

We use $L^2(\phi_d)$ as our underlying function space with Gaussian measure $\phi_d = (\frac{1}{2\pi})^{\frac{d}{2}} e^{-\frac{|\mathbf{x}|^2}{2}}$. $\|f\|^2 = \int_{R^d} f^2 \phi_d d\mathbf{x}$. The smoothness classes characterise the rates of convergence. Let $\Delta_d$ be the Laplacian operator and $\Delta_d^*$ be the Laplacian-Beltrami operator (Muller, 1966). The smoothness classes can be defined as:

**Definition 1** *The function $f \in L^2(\phi_d)$ will be said to have Cartesian smoothness of order $p$ if it has $p$ derivatives and these derivatives are all in $L^2(\phi_d)$. It will be said to have angular smoothness of order $q$ if $\Delta_d^{*q} f \in L^2(\phi_d)$. It will be said to have Laplacian smoothness of order $r$ if $\Delta_d^r f \in L^2(\phi_d)$. Let $\mathcal{F}_p$ be the class of functions with Cartesian smoothness $p$, $\mathcal{A}_{pq}$ be the class of functions with Cartesian smoothness $p$ and angular smoothness $q$ and $\mathcal{L}_{pr}$ be the class of functions with Cartesian smoothness $p$ and Laplacian smoothness $r$.*

We derive an orthogonal basis in $L^2(\phi_d)$ from the eigenfunctions of a self-adjoint operator. The basis element is defined as:

$$J_{njm}(\mathbf{x}) = \gamma_m \underbrace{r^n S_{nj}(\xi)}_{harmonic} \underbrace{L_m^\alpha(\frac{r^2}{2})}_{radial} \tag{3}$$

where $\mathbf{x} = r\xi$, $n = 0, ..., \infty, m = 0, ..., \infty, j = 1, ..., N(d,n), \gamma_m = (-2)^m m!, \alpha = n + \frac{d-2}{2}$. $S_{nj}(\xi)$ are linearly independent spherical harmonics of degree $n$ in $d$ dimensions (Muller, 1966). $L_m^\alpha(\frac{r^2}{2})$ is a Laguerre polynomial. The advantage of the basis comes from its representation as a product of a spherical harmonic and a radial polynomial. Specifically $J_{0jm}$ is a radial polynomial for $n = 0$ and $J_{nj0}$ is a harmonic polynomial for $m = 0$. Figure 2(a),(b) show a radial basis element and a harmonic basis element when $n + 2m = 8$. The basis element $J_{njm}$ has an orthogonality:

$$E(J_{njm}(r,\xi)J_{kil}(r,\xi)) = \delta_n^k \delta_j^i \delta_m^l \frac{2^{n+2m-1}}{\pi^{\frac{d}{2}}} \Gamma(m + n + \frac{d}{2}) \Gamma(m+1) \tag{4}$$

where $E$ denotes expectation with respect to $\phi_d$. Since it is a basis in $L^2(\phi_d)$, any function $f \in L^2(\phi_d)$ has an expansion in terms of basis elements $J_{njm}$

$$f = \sum_{n,j,m} c_{njm} J_{njm} \tag{5}$$

The circular harmonic $e^{in\theta}$ is a special case of the spherical harmonic $S_{nj}(\xi)$. In two dimensions, $d = 2$, $N(d,n) = 2$ and $\mathbf{x} = (r\cos\theta, r\sin\theta)$. The spherical harmonic $S_{nj}(\xi)$ can be reduced to the following: $S_{n1}(\xi) = \frac{1}{\sqrt{\pi}}\cos n\theta$, $S_{n2}(\xi) = \frac{1}{\sqrt{\pi}}\sin n\theta$, which is the circular harmonic.

Smoothness classes can also be defined qualitatively from expansions of functions in terms of basis elements $J_{njm}$. Since $\|f\|^2 = \sum c_{njm}^2 J_{njm}^2$, one can think of $p_{njm}(f) = \frac{c_{njm}^2 J_{njm}^2}{\sum c_{njm}^2 J_{njm}^2}$ as representing the distribution of energy in $f$ among the different modes of oscillation $J_{njm}$. If $f$ is cartesian smooth, $p_{njm}(f)$ peaks around small $n + 2m$. If $f$ is angular smooth, $p_{njm}(f)$ peaks around small $n$. If $f$ is Laplacian smooth, $p_{njm}(f)$ peaks around small $m$. To explain why PPR works better on angular smooth functions than on Laplacian smooth functions, we first examine how to represent these $L^2(\phi_d)$ basis elements systematically in terms of ridge functions and then use the expansion (5) to derive a error bound of RA for general smooth functions.

## 3    CONTINUOUS RIDGE FUNCTION SCHEMES

There exists a continuous ridge function representation for any function $f(\mathbf{x}) \in L^2(\phi_d)$ which is an integral of ridge functions through all possible directions.

$$f(\mathbf{x}) = \int_{\Omega_d} g(\mathbf{x}^T\eta, \eta) d\omega_d(\eta). \tag{6}$$

This works intuitively because any object is determined by any infinite set of radiographs. More precisely, any function $f(\mathbf{x}) \in L^2(\phi_d)$ can be approximated arbitrarily well by a linear combination of ridge functions $\sum_k g(\mathbf{x}^T\eta_k, \eta_k)$ provided infinitely many combination units (Jones, 1987). As $k \to \infty$, we have (6). The natural discrete approximation to (6) has the form: $f_n(\mathbf{x}) = \sum_{j=1}^n w_j g(\mathbf{x}^T\eta_j, \eta_j)$, which becomes the usual PPR (2). We proved a continuous ridge function representation of basis elements $J_{njm}$ which is shown in Lemma 1.

**Lemma 1** *The continuous ridge function representaion of $J_{njm}$ is:*

$$J_{njm}(\mathbf{x}) = \lambda_{nmd} \int_{\Omega_d} H_{n+2m}(\eta^T\mathbf{x}) S_{nj}(\eta) d\omega_d(\eta) \tag{7}$$

*where $\lambda_{nmd}$ is a constant and $H_{n+2m}(x)$ is a Hermite polynomial.*

Therefore any function $f \in L^2(\phi_d)$ has a continuous ridge function representation (6) with

$$g(\mathbf{x}^T\eta, \eta) = \sum c_{njm}\lambda_{nmd}H_{n+2m}(\mathbf{x}^T\eta)S_{nj}(\eta) \tag{8}$$

Gaussian quadrature and Monte Carlo integration schemes can be used to discretize (6).

## 4    GAUSSIAN QUADRATURE

Since $\int_{\Omega_d} g(\mathbf{x}^T\eta, \eta) d\omega_d(\eta) = \int_{\Omega_{d-1}} \int_{-1}^{1} g(\mathbf{x}^T\eta, \eta)(1 - t_{d-1}^2)^{\frac{d-3}{2}} dt_{d-1} d\omega_{d-1}(\eta_{d-1})$, simple product rules using Gaussian quadrature formulae can be used here. $t_{ij}, i =$

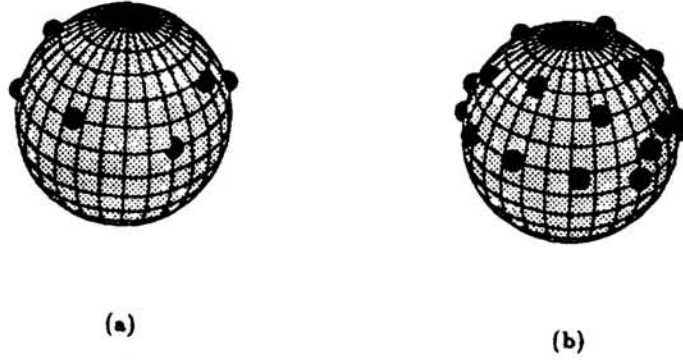

(a)

(b)

Figure 3: Directions (a) for a radial polynomial (b) for a harmonic polynomial

$d-1, ..., 1, j = 1, ..., n$ are zeros of orthogonal polynomials with weights $(1 - t^2)^{\frac{i-2}{3}}$.
$N = n^{d-1}$ points on the unit sphere $\Omega_d$ can be formed using $t_{ij}$ through

$$\eta = \begin{bmatrix} \sqrt{1 - t_{d-1}^2} \cdots \sqrt{1 - t_1^2} \\ \sqrt{1 - t_{d-1}^2} \cdots t_1 \\ \cdots \\ t_{d-1} \end{bmatrix} \qquad (9)$$

If $g(\mathbf{x}^T \eta, \eta)$ is a polynomial of degree at most $2n - 1$ (in terms of $t_1, ..., t_{d-1}$), then $N = n^{d-1}$ points (directions) are sufficient to represent the integral exactly. This can be demonstrated with two examples by taking $d = 3$.

Example 1: a radial function

$$x^4 + y^4 + z^4 + 2x^2y^2 + 2x^2z^2 + 2y^2z^2 = c_1 \int_{\Omega_3} (\mathbf{x}^T \eta)^4 d\omega_3(\eta) \qquad (10)$$

$d = 3, n = 3$. $n^2 = 9$ directions from (9) are sufficient to represent this polynomial with $t_2 = 0, \sqrt{\frac{3}{5}}, -\sqrt{\frac{3}{5}}$ (zeros of a degree 3 Legendre polynomial) and $t_1 = 0, \frac{\sqrt{3}}{2}, -\frac{\sqrt{3}}{2}$ ( zeros of a degree 3 Tschebyscheff polynomial). More directions are needed to represent a harmonic function with exactly the same number of terms of monomials but with different coefficients.

Example 2: a harmonic function

$$\frac{1}{8}(8x^4 + 3y^4 + 3z^4 - 24x^2y^2 - 24x^2z^2 + 6y^2z^2) = c_2 \int_{\Omega_3} (\mathbf{x}^T \eta)^4 S_{4j}(\eta) d\omega_3(\eta) \quad (11)$$

where $S_{4j}(\eta) = \frac{1}{8}(35t^4 - 30t^2 + 3), \eta = t\epsilon_3 + \sqrt{1 - t^2}\eta_2$. $n = 5, n^2 = 25$ directions from (9) are sufficient to represent the polynomial with $t_2 = 0, 0.90618, -0.90618, 0.53847, -0.53847$ and $t_1 = \cos \frac{2j-1}{10}\pi, j = 1, ..., 5$. The distribution of these directions on a unit sphere are shown in Figure 3(a) and (b). In general, $N = (n + m + 1)^{d-1}$ directions are sufficient to represent $J_{njm}$ exactly by using zeros of orthogonal polynomials. If $p = n + 2m$ (the degree of the basis) is fixed, $N = (p - m + 1)^{d-1} = (\frac{p+n}{2} + 1)^{d-1}$ is minimised when $n = 0$ which corresponds to the radial basis element. $N$ is maximised when $m = 0$ which is the harmonic element. Using definitions of smoothness classes in Section 2. we can show that ridge function approximation works better on angular smooth functions. The basic result is as follows:

**Theorem 1** *$f \in \mathcal{A}_{pq}$, let $R_N f$ denote a sum of ridge functions which best approximate $f$ by using a set of directions generated by zeros of orthogonal polynomials. Then*

$$E_N = \|R_N f - f\|^2_{\mathcal{A}_{pq}} \leq B_p N^{-\frac{p}{d-1}} + B_q N^{-\frac{q}{d-1}} \qquad (12)$$

This error bound says that ridge function approximation does take advantage of angular smoothness. Radial functions are the most angular smooth functions with $q = +\infty$ and harmonic functions are the least angular smooth functions when the Cartesian smoothness $p$ is fixed. Therefore ridge function approximation works better on angular smooth functions than on Laplacian smooth functions. Radial and harmonic functions are the two extreme cases.

## 5 UNIFORMLY DISTRIBUTED DIRECTIONS ON $\Omega_d$

Instead of using directions from zeros of orthogonal polynomials, N uniformly distributed directions on $\Omega_d$ is an alternative to generalizing equispaced directions. This is a Monte Carlo type of integration scheme on $\Omega_d$.

To approximate the integral (7), $N$ uniformly distributed directions $\eta_1$, $\eta_2$, ......, $\eta_N$ on $\Omega_d$ drawn from the density $f(\eta) = 1/\omega_d$ on $\Omega_d$ are used:

$$\hat{J}_{njm}(\mathbf{x}) = \frac{\omega_d}{N} \lambda_{nmd} \sum_{k=1}^{N} H_{n+2m}(\mathbf{x}^T \eta_k) S_{nj}(\eta_k) \qquad (13)$$

The mean value for $\hat{J}_{njm}(\mathbf{x})$ is

$$m_N(\mathbf{x}) = \frac{\omega_d}{N} \lambda_{nmd} \sum_{k=1}^{N} \int_{\Omega_d} H_{n+2m}(\mathbf{x}^T \eta_k) S_{nj}(\eta_k) \frac{1}{\omega_d} d\omega_d(\eta_k) = J_{njm}(\mathbf{x}) \qquad (14)$$

The variance is

$$\sigma_N^2(\mathbf{x}) = \frac{\sigma^2(\mathbf{x})}{N} \qquad (15)$$

where $\sigma^2(\mathbf{x}) = \int_{\Omega_d} \left[ \lambda_{nmd} \omega_d H_{n+2m}(\mathbf{x}^T \eta) S_{nj}(\eta) - J_{njm}(\mathbf{x}) \right]^2 \frac{1}{\omega_d} d\omega_d(\eta)$. Therefore $\hat{J}_{njm}(\mathbf{x})$ approximates $J_{njm}(\mathbf{x})$ with a rate $\sigma(\mathbf{x})/\sqrt{N}$. The difference between a radial basis and a harmonic basis is $\sigma(\mathbf{x})$. Let us average $\sigma(\mathbf{x})$ with respect to $\phi_d$:

$$\|\sigma(\mathbf{x})\|^2 = \|J_{njm}\|^2 \left[ \frac{\|J_{njm}\|^2 \omega_d}{\Gamma(n+2m+1)} - 1 \right] = \|J_{njm}\|^2 \alpha_{njm} \qquad (16)$$

For a fixed $n + 2m = p$, $\|\sigma(\mathbf{x})\|^2$ is minimized at $n = 0$ (a radial element) and maximized at $m = 0$ (a harmonic element).

The same justification can be done for a general function $f \in L^2(\phi_d)$ with (6) and (8). A RA scheme is:

$$\hat{f}(\mathbf{x}) = \frac{\omega_d}{N} \sum_{k=1}^{N} g(\mathbf{x}^T \eta_k, \eta_k)$$

$m_N(\mathbf{x}) = f(\mathbf{x})$, $\sigma_N^2(\mathbf{x}) = \frac{\sigma^2(\mathbf{x})}{N}$, $\sigma^2(\mathbf{x}) = \int_{\Omega_d} \omega_d g^2(\mathbf{x}^T\eta, \eta) d\omega_d(\eta) - f^2(\mathbf{x})$ and $\|\sigma(\mathbf{x})\|^2 = \sum c_{njm}^2 \|J_{njm}\|^2 \alpha_{njm}$. Since $\alpha_{njm}$ is small when $n$ is small, large when $m$ is small and recall the distribution $p_{njm}(f)$ in Section 3, $\|\sigma\|^2/\|f\|^2$ is small when $f$ is smooth. Among these smooth functions, if $f$ is angular smooth, $\|\sigma\|^2/\|f\|^2$ is smaller than that if $f$ is Laplacian smooth. The RMS error convergence rate $\frac{\|\sigma_N\|}{\|f\|} = \frac{\|\sigma\|}{\|f\|\sqrt{N}}$ is consequently smaller for $f$ being angular smooth than for $f$ being Laplacian smooth. But both rates are $O(N^{-\frac{1}{2}})$ no matter what class the underlying function belongs to. The difference is the constant which is related to the distribution of energy in $f$ among the different modes of oscillations (angular or Laplacian). The radial and harmonic functions are two extremes.

## 6     THE CURSE OF DIMENSIONALITY IN RA

Generally, if $N$ directions $\eta_1$, $\eta_2$, ......, $\eta_N$, on $\Omega_d$ drawn from any distribution $p(\eta)$ on the sphere $\Omega_d$ to approximate (6)

$$\hat{f}(\mathbf{x}) = \frac{1}{N} \sum_{k=1}^{N} \frac{g(\mathbf{x}^T\eta_k, \eta_k)}{p(\eta_k)}) \tag{17}$$

$m_N = f(\mathbf{x})$ $\sigma_N^2 = \frac{\sigma^2}{N}$ where $\sigma^2(\mathbf{x}) = \int_{\Omega_d} \frac{1}{p(\eta)} g^2(\mathbf{x}^T\eta, \eta) d\omega_d(\eta) - f^2(\mathbf{x})$. And

$$\|\sigma(\mathbf{x})\|^2 = \int_{\Omega_d} \frac{1}{p(\eta)} \|g_\eta\|^2 d\omega_d(\eta) - \|f\|^2 = c_f \tag{18}$$

Then

$$\|\hat{f}(\mathbf{x}) - f(\mathbf{x})\|^2 \leq \frac{\|\sigma(\mathbf{x})\|^2}{N} = \frac{c_f}{N} \tag{19}$$

That is, $\hat{f}(\mathbf{x}) \to f(\mathbf{x})$ with a rate $O(N^{-\frac{1}{2}})$. Equation (19) shows that there is no curse of dimensionality if a ridge function approximation scheme (17) is used for $f(\mathbf{x})$. The same conclusion can be drawn when sigmoidal hidden unit function neural networks are applied to Barron's class of underlying function (Barron, 1991). But our function class here is the function class that can be represented by a continuous ridge function (6), which is a much larger function class than Barron's. Any function $f \in L^2(\phi_d)$ has a representation (6)(Section 4). Therefore, for any function $f \in L^2(\phi_d)$, there exists a node function $g(\mathbf{x}^T\eta, \eta)$ and related ridge function approximation scheme (17) to approximate $f(\mathbf{x})$ with a rate $O(N^{-\frac{1}{2}})$, which has no curse of dimensionality. In other words, if we are allowed to choose a node function $g(\mathbf{x}^T\eta, \eta)$ according to the property of data, which is the characteristic of PPR, then ridge function approximation scheme can avoid the curse of dimensionality. That is a generalization of Barron's result that the curse of dimensionality goes away if certain types of node function (*e.g.*, *cos* and $\sigma$) are considered.

The smoothness of a underlying function determines the size of the constant $c_f$. As shown in the previous section, if $p(\eta) = 1/\omega_d$ (*i.e.*, uniformly distributed directions), then angular smooth functions have smaller $c_f$ than Laplacian smooth functions do. Choosing different $p(\eta)$ does not change this conclusion. But a properly chosen $p(\eta)$ reduces $c_f$ in general. If $f(\mathbf{x})$ is smooth enough, the node function $g(\mathbf{x}^T\eta, \eta)$ can be

computed from the Radon transform $R_\eta f$ of $f$ in the direction $\eta$ which is defined as

$$R_\eta f(\mathbf{x}^T \eta = s) = \int_{R^{d-1}} f(\mathbf{x}^T \eta \eta + \mathbf{x}_{d-1} \eta_{d-1}) d\mathbf{x}_{d-1} \qquad (20)$$

and we proved: $g(\mathbf{x}^T \eta, \eta) = \mathcal{F}^{-1}(F_\eta(t)|t|^{d-1}) |_{t=\mathbf{x}^T\eta}$, where $F_\eta(t)$ is the Fourier transform of $R_\eta f(s)$ and $\mathcal{F}^{-1}$ denotes the inverse Fourier transform. In practice, learning $g(\mathbf{x}^T \eta, \eta)$ is usually replaced by a smoothing step which seeks a one dimensional function to fit $\mathbf{x}^T \eta$ best to the residual in this direction (Friedman and Stuetzle, 1981, Zhao and Atkeson, 1991).

# 7   CONCLUSION

As we showed, PPR works better on angular smooth function than on Laplacian smooth functions by discretizing a continuous ridge function representation. PPR can avoid the curse of dimensionality by learning node functions from data.

**Acknowledgments**

Support was provided under Air Force Office of Scientific Research grant AFOSR-89-0500. Support for CGA was provided by a National Science Foundation Presidential Young Investigator Award, an Alfred P. Sloan Research Fellowship, and the W. M. Keck Foundation Associate Professorship in Biomedical Engineering. Special thanks goes to Prof. Zhengfang Zhou and Prof. Peter Huber at Math Dept. in MIT, who provided useful discussions.

**References**

**Barron, A. R. and Barron, R. L.** (1988) "Statistical Learning Networks: A Unifying View." *Computing Science and Statistics: Proceedings of 20th Symposium on the Interface.* Ed Wegman, editor, Amer. Statist. Assoc., Washington, D. C., 192-203.

**Barron, A. R.** (1991) "Universal Approximation Bounds for Superpositions of A Sigmoidal Function". TR. 58. Dept. of Stat., Univ. of Illinois at Urbana-Champaign.

**Donoho, D. L. and Johnstone, I.** (1989). "Projection-based Approximation, and Duality with Kernel Methods". *Ann. Statist., 17, 58-106.*

**Diaconis, P. and Shahshahani, M.** (1984) "On Non-linear Functions of Linear Combinations", *SIAM J. Sci. Stat. Compt. 5, 175-191.*

**Friedman, J. H. and Stuetzle, W.** (1981) "Projection Pursuit Regression". *J. Amer. Stat. Assoc.,* 76, 817-823.

**Huber, P. J.** (1985) "Projection Pursuit (with discussion)", *Ann. Statist., 13, 435-475.*

**Jones, L.** (1987) "On A Conjecture of Huber Concerning the Convergence of Projection Pursuit Regression". *Ann. Statist., 15, 880-882.*

**Muller, C.** (1966), *Spherical Harmonics.* Lecture Notes in Mathematics, no.17.

**Zhao, Y. and C. G. Atkeson** (1991) "Projection Pursuit Learning", Proc. IJCNN-91-SEATTLE.